# *Real-Time Computer Vision and Robotics Using Analog VLSI Circuits*

Christof Koch     Wyeth Bair     John G. Harris     Timothy Horiuchi
Andrew Hsu     Jin Luo
Computation and Neural Systems Program
Caltech 216-76
Pasadena, CA 91125

## ABSTRACT

The long-term goal of our laboratory is the development of analog resistive network-based VLSI implementations of early and intermediate vision algorithms. We demonstrate an experimental circuit for smoothing and segmenting noisy and sparse depth data using the resistive fuse and a 1-D edge-detection circuit for computing zero-crossings using two resistive grids with different space-constants. To demonstrate the robustness of our algorithms and of the fabricated analog CMOS VLSI chips, we are mounting these circuits onto small mobile vehicles operating in a real-time, laboratory environment.

## 1   INTRODUCTION

A large number of computer vision algorithms for finding intensity edges, computing motion, depth, and color, and recovering the 3-D shapes of objects have been developed within the framework of minimizing an associated "energy" functional. Such a variational formalism is attractive because it allows *a priori* constraints to be explicitly stated. The single most important constraint is that the physical processes underlying image formation, such as depth, orientation and surface reflectance, change slowly in space. For instance, the depths of neighboring points on a surface are usually very similar. Standard regularization algorithms embody this smoothness constraint and lead to quadratic variational functionals with a unique global minimum (Poggio, Torre, and Koch, 1985). These quadratic functionals

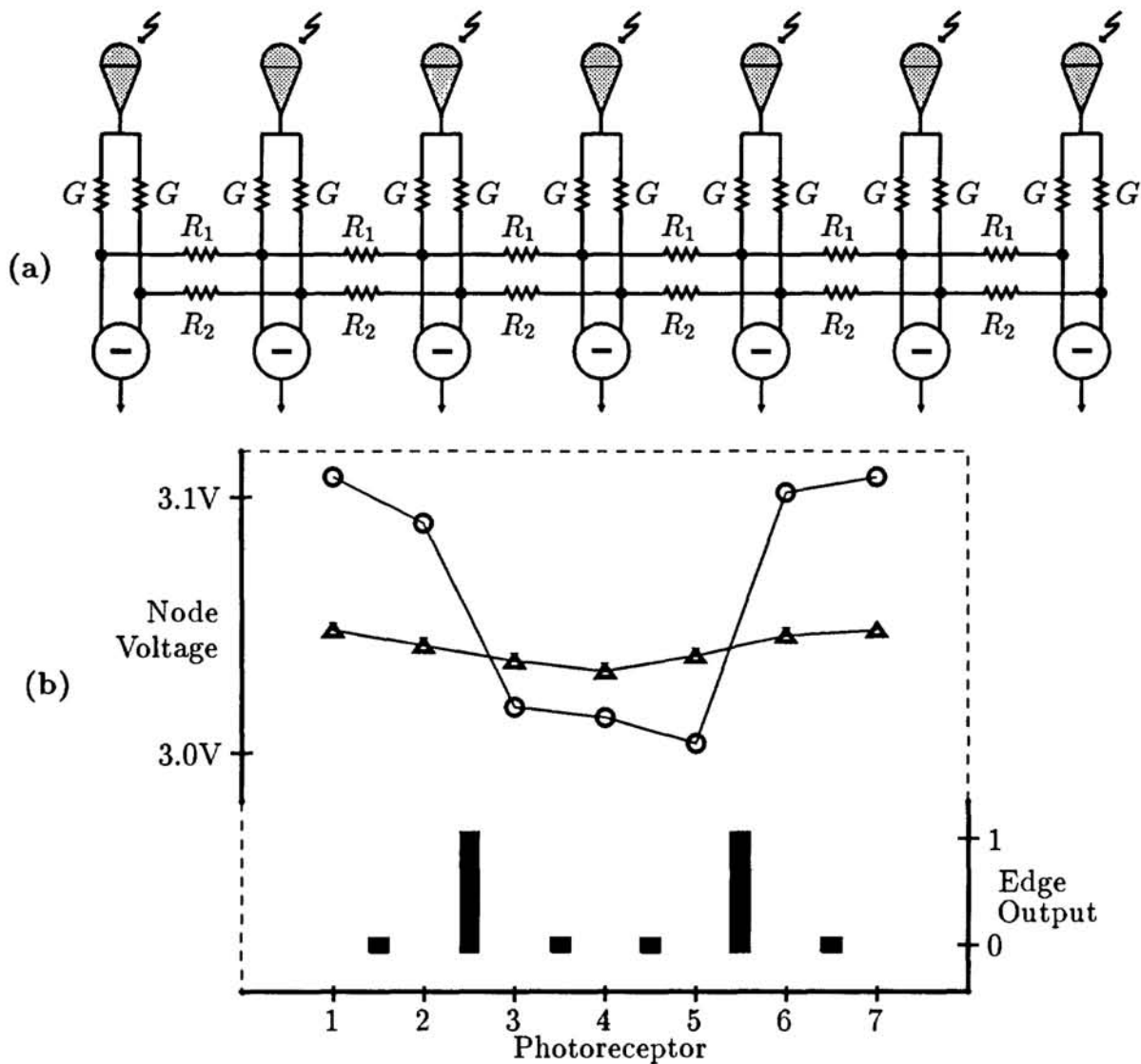

**Figure 1:** (a) shows the schematic of the zero-crossing chip. The phototransistors logarithmically map light intensity to voltages that are applied via a conductance $G$ onto the nodes of two linear resistive networks. The network resistances $R_1$ and $R_2$ can be arbitrarily adjusted to achieve different space-constants. Transconductance amplifiers compute the difference of the smoothed network node voltages and report a current proportional to that difference. The sign of current then drives exclusive-or circuitry (not shown) between each pair of neighboring pixels. The final output is a binary signal indicating the positions of the zero-crossings. The linear network resistances have been implemented using Mead's saturating resistor circuit (Mead, 1989), and the vertical resistors are implemented with transconductance followers. (b) shows the measured response of a seven-pixel version of the chip to a bright background with a shadow cast across the middle three photoreceptors. The circles and triangles show the node voltages on the resistive networks with the smaller and larger space-constants, respectively. Edges are indicated by the binary output (bar chart at bottom) corresponding to the locations of zero-crossings.

can be mapped onto linear resistive networks, such that the stationary voltage distribution, corresponding to the state of least power dissipation, is equivalent to the solution of the variational functional (Horn, 1974; Poggio and Koch, 1985). Smoothness breaks down, however, at discontinuities caused by occlusions or differences in the physical processes underlying image formation (e.g., different surface reflectance properties). Detecting these discontinuities becomes crucial, not only because otherwise smoothness is incorrectly applied but also because the locations of discontinuities are often required for further image analysis and understanding. We describe two different approaches for finding discontinuities in early vision: (1) a 1-D edge-detection circuit for computing zero-crossings using two resistive grids with different space-constants, and (2) a 20 by 20 pixel circuit for smoothing and segmenting noisy and sparse depth data using the resistive fuse.

Finally, while successfully demonstrating a highly integrated circuit on a stationary laboratory bench under controlled conditions is already a tremendous success, this is not the environment in which we ultimately intend them to be used. The jump from a sterile, well-controlled, and predictable environment such as that of the laboratory bench to a noisy and physically demanding environment of a mobile robot can often spell out the true limits of a circuit's robustness. In order to demonstrate the robustness and real-time performance of these circuits, we have mounted two such chips onto small toy vehicles.

## 2    AN EDGE DETECTION CIRCUIT

The zero-crossings of the Laplacian of the Gaussian, $\nabla^2 G$, are often used for detecting edges. Marr and Hildreth (1980) discovered that the Mexican-hat shape of the $\nabla^2 G$ operator can be approximated by the difference of two Gaussians (DOG). In this spirit, we have built a chip that takes the difference of two resistive-network smoothings of photoreceptor input and finds the resulting zero-crossings. The Green's function of the resistive network, a decaying exponential, differs from the Gaussian, but simulations with digitized camera images have shown that the difference of exponentials (DOE) gives results nearly as good as the DOG. Furthermore, resistive nets have a natural implementation in silicon, while implementing the Gaussian is cumbersome.

The circuit, Figure 1a, uses two independent resistive networks to smooth the voltages supplied by logarithmic photoreceptors. The voltages on the two networks are subtracted and exclusive-or circuitry (not shown) is used to detect zero-crossings. In order to facilitate thresholding of edges, an additional current is computed at each node indicating the strength of the zero-crossing. This is particularly important for robust real-world performance where there will be many small (in magnitude of slope) zero-crossings due to noise. Figure 1b shows the measured response of a seven-pixel version of the chip to a bright background with a shadow cast across the middle three photoreceptors. Subtracting the two network voltage traces shown at the top, we find two zero-crossings, which the chip correctly identifies in the binary output shown at the bottom.

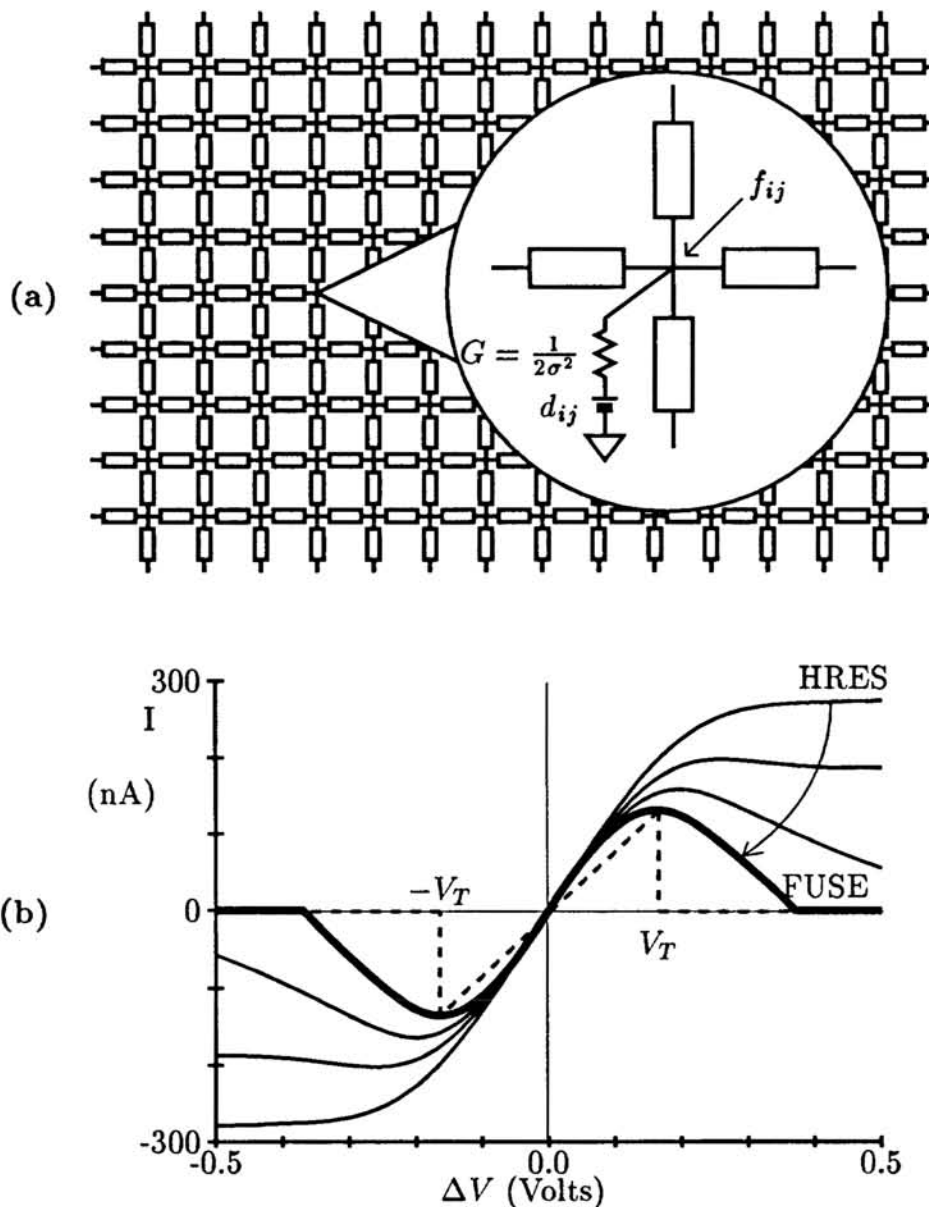

**Figure 2:** **(a)** Schematic diagram for the 20 by 20 pixel surface interpolation and smoothing chip. A rectangular mesh of resistive fuse elements (shown as rectangles) provide the smoothing and segmentation ability of the network. The data are given as battery values $d_{ij}$ with the conductance $G$ connecting the battery to the grid set to $G = 1/2\sigma^2$, where $\sigma^2$ is the variance of the additive Gaussian noise assumed to corrupt the data. **(b)** Measured current-voltage relationship for different settings of the resistive fuse. For a voltage of less than $V_T$ across this two-terminal device, the circuit acts as a resistor with conductance $\lambda$. Above $V_T$, the current is either abruptly set to zero (binary fuse) or smoothly goes to zero (analog fuse). We can continuously vary the I-V curve from the hyperbolic tangent of Mead's saturating resistor (HRES) to that of an analog fuse (Fig. 2b), effectively implementing a continuation method for minimizing the non-convex functional. The I-V curve of a binary fuse is also illustrated.

## 3    A CIRCUIT FOR SMOOTHING AND SEGMENTING

Many researchers have extended regularization theory to include discontinuities. Let us consider the problem of interpolating noisy and sparse 1-D data (the 2-D generalization is straightforward), where the depth data $d_i$ is given on a discrete grid. Associated with each lattice point is the value of the recovered surface $f_i$ and a binary line discontinuity $\ell_i$. When the surface is expected to be smooth (with a first-order, membrane-type stabilizer) except at isolated discontinuities, the functional to be minimized is given by:

$$J(f, \ell) = \lambda \sum_i (f_{i+1} - f_i)^2 (1 - \ell_i) + \frac{1}{2\sigma^2} \sum_i (d_i - f_i)^2 + \alpha \sum_i \ell_i \qquad (1)$$

where $\sigma^2$ is the variance of the additive Gaussian noise process assumed to corrupt the data $d_i$, and $\lambda$ and $\alpha$ are free parameters. The first term implements the piecewise smooth constraint: if all variables, with the exception of $f_i, f_{i+1}$, and $\ell_i$, are held fixed and $\lambda(f_{i+1} - f_i)^2 < \alpha$, it is "cheaper" to pay the price $\lambda(f_{i+1} - f_i)^2$ and set $\ell_i = 0$ than to pay the larger price $\alpha$; if the gradient becomes too steep, $\ell_i = 1$, and the surface is segmented at that location. The second term, with the sum only including those locations $i$ where data exist, forces the surface $f$ to be close to the measured data $d$. How close depends on the estimated magnitude of the noise, in this case on $\sigma^2$. The final surface $f$ is the one that best satisfies the conflicting demands of piecewise smoothness and fidelity on the measured data.

To minimize the 2-D generalization of eq. (1), we map the functional $J$ onto the circuit shown in Fig. 2a such that the stationary voltage at every gridpoint then corresponds to $f_{ij}$. The cost functional $J$ is interpreted as electrical co-content, the generalization of power for nonlinear networks. We designed a two-terminal nonlinear device, which we call a resistive fuse, to implement piecewise smoothness (Fig. 2b). If the magnitude of the voltage drop across the device is less than $V_T = (\alpha/\lambda)^{1/2}$, the fuse acts as a linear resistor with conductance $\lambda$. If $V_T$ is exceeded, however, the fuse breaks and the current goes to zero. The operation of the fuse is fully reversible. We built a 20 by 20 pixel fuse network chip and show its segmentation and smoothing performance in Figure 3.

## 4    AUTONOMOUS VEHICLES

Our goal—beyond the design and fabrication of analog resistive-network chips—is to build mobile testbeds for the evaluation of chips as well as to provide a systems perspective on the usefulness of certain vision algorithms. Due to the small size and power requirements of these chips, it is possible to utilize the vast resource of commercially available toy vehicles. The advantages of toy cars over robotic vehicles built for research are their low cost, ease of modification, high power-to-weight ratio, availability, and inherent robustness to the real-world. Accordingly, we integrated two analog resistive-network chips designed and built in Mead's laboratory onto small toy cars controlled by a digital microprocessor (see Figure 4).

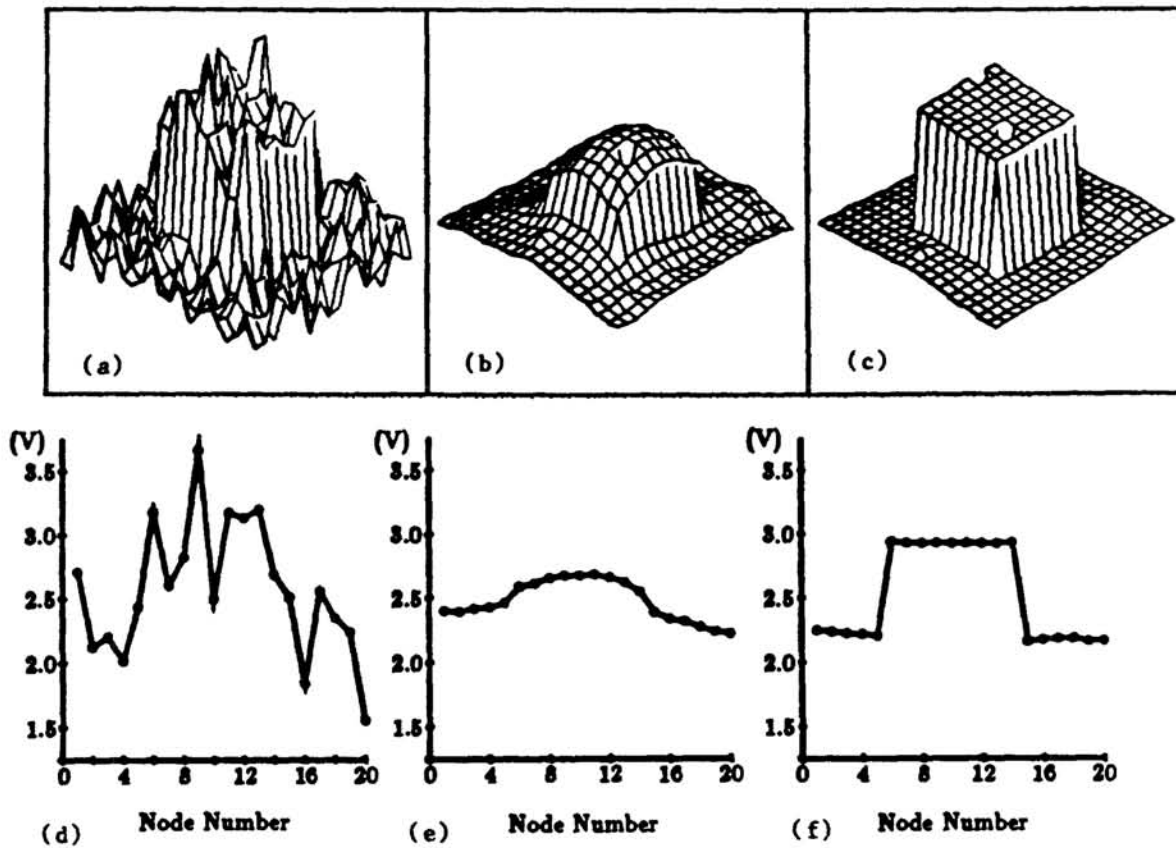

**Figure 3:** Experimental data from the fuse network chip. We use as input data a tower (corresponding to $d_{ij} = 3.0\ V$) rising from a plane (corresponding to $2.0\ V$) with superimposed Gaussian noise. **(a)** shows the input with the variance of the noise set to $0.2\ V$, **(b)** the voltage output using the fuse configured as a saturating resistance, and **(c)** the output when the fuse elements are activated. **(d)**, **(e)**, and **(f)** illustrate the same behavior along a horizontal slice across the chip for $\sigma^2 = 0.4\ V$. We used a hardware deterministic algorithm of varying the fuse I-V curve of the saturating resistor to that of the analog fuse (following the arrow in Fig. 2b) as well as increasing the conductance $\lambda$. This algorithm is closely related to other deterministic approximations based on continuation methods or a Mean Field Theory approach (Koch, Marroquin, and Yuille, 1986; Blake and Zisserman, 1987; Geiger and Girosi, 1989). Notice that the amplitude of the noise in the last case (40% of the amplitude of the voltage step) is so large that a single filtering step on the input **(d)** will fail to detect the tower. Cooperativity and hysteresis are required for optimal performance. Notice the "bad" pixel in the middle of the tower (in **c**). Its effect is localized, however, to a single element.